# Bangs, Clicks, Snaps, Thuds and Whacks: an Architecture for Acoustic Transient Processing

**Fernando J. Pineda**[1]
fernando.pineda@jhuapl.edu

**Gert Cauwenberghs**[2]
gert@jhunix.hcf.jhu.edu

**R. Timothy Edwards**[2]
tim@bach.ece.jhu.edu

[1]The Applied Physics Laboratory
The Johns Hopkins University
Laurel, Maryland 20723-6099

[2]Dept. of Electrical and Computer Engineering
The Johns Hopkins University
34th and Charles Streets
Baltimore Maryland 21218

## ABSTRACT

We propose a neuromorphic architecture for real-time processing of acoustic transients in analog VLSI. We show how judicious normalization of a time-frequency signal allows an elegant and robust implementation of a correlation algorithm. The algorithm uses binary multiplexing instead of analog-analog multiplication. This removes the need for analog storage and analog-multiplication. Simulations show that the resulting algorithm has the same out-of-sample classification performance (~93% correct) as a baseline template-matching algorithm.

## 1  INTRODUCTION

We report progress towards our long-term goal of developing low-cost, low-power, low-complexity analog-VLSI processors for real-time applications. We propose a neuromorphic architecture for acoustic processing in analog VLSI. The characteristics of the architecture are explored by using simulations and real-world acoustic transients. We use acoustic transients in our experiments because information in the form of acoustic transients pervades the natural world. Insects, birds, and mammals (especially marine mammals) all employ acoustic signals with rich transient structure. Human speech, is largely composed of transients and speech recognizers based on transients can perform as well as recognizers based on phonemes (Morgan, Bourlard,Greenberg, Hermansky, and Wu, 1995). Machines also generate transients as they change state and as they wear down. Transients can be used to diagnose wear and abnormal conditions in machines.

In this paper, we consider how algorithmic choices that do not influence classification performance, make an initially difficult-to-implement algorithm, practical to implement. In particular, we present a practical architecture for performing real-time recognition of acoustic transients via a correlation-based algorithm. Correlation in analog VLSI poses two fundamental implementation challenges. First, there is the problem of template storage, second, there is the problem of accurate analog multiplication. Both problems can be solved by building sufficiently complex circuits. This solution is generally unsatisfactory because the resulting processors must have less area and consume less power than their digital counterparts in order to be competitive. Another solution to the storage problem is to employ novel floating gate devices. At present such devices can store analog values for years without significant degradation. Moreover, this approach can result in very compact, yet computationally complex devices. On the other hand, programming floating gate devices is not so straight-forward. It is relatively slow, it requires high voltage and it degrades the floating gate each time it is reprogrammed. Our "solution" is to side-step the problem completely and to develop an algorithmic solution that requires neither analog storage nor analog multiplication. Such an approach is attractive because it is both biologically plausible and electronically efficient. We demonstrate that a high level of classification performance on a real-world data set is achievable with no measurable loss of performance, compared to a baseline correlation algorithm.

The acoustic transients used in our experiments were collected by K. Ryals and D. Steigerwald and are described in (Pineda, Ryals, Steigerwald and Furth, 1995). These transients consist of isolated Bangs, Claps, Clicks, Cracks, Dinks, Pings, Pops, Slaps, Smacks, Snaps, Thuds and Whacks that were recorded on DAT tape in an office environment. The ambient noise level was uncontrolled, but typical of a single-occupant office. Approximately 221 transients comprising 10 classes were collected. Most of the energy in one of our typical transients is dissipated in the first 10 ms. The remaining energy is dissipated over the course of approximately 100 ms. The transients had durations of approximately 20-100 ms. There was considerable in-class and extra-class variability in duration. The duration of a transient was determined automatically by a segmentation algorithm described below. The segmentation algorithm was also used to align the templates in the correlation calculations.

## 2 THE BASELINE ALGORITHM

The baseline classification algorithm and its performance is described in Pineda, et al. (1995). Here we summarize only its most salient features. Like many biologically motivated acoustic processing algorithms, the preprocessing steps include time-frequency analysis, rectification, smoothing and compression via a nonlinearity (e.g. Yang, Wang and Shamma, 1992). Classification is performed by correlation against a template that represents a particular class. In addition , there is a "training" step which is required to create the templates. This step is described in the "correlation" section below. We turn now to a more detailed description of each processing step.

A. *Time-frequency Analysis:* Time-frequency analysis for the baseline algorithm and the simulations performed in this work, was performed by an ultra-low power (5.5 mW) analog VLSI filter bank intended to mimic the processing performed by the mammalian cochlea (Furth, Kumar, Andreou and Goldstein, 1994). This real-time device creates a time-frequency representation that would ordinarily require hours of computation on a

high-speed workstation. More complete descriptions can be found in the references. The time-frequency representation produced by the filter bank is qualitatively similar to that produced by a wavelet transformation. The center frequencies and Q-factors of each channel are uniformly spaced in log space. The low frequency channel is tuned to a center frequency of 100 Hz and Q-factor of 1.0, while the high frequency channel is tuned to a center frequency of 6000 Hz and Q-factor 3.5. There are 31 output channels. The 31-channel cochlear output was digitized and stored on disk at a raw rate of 256K samples per second. This raw rate was distributed over 32 channels, at rates appropriate for each channel (six rates were used, 1 kHz for the lowest frequency channels up to 32 kHz for the highest-frequency channels and the unfiltered channel).

***B. Segmentation:*** Both the template calculation and the classification algorithm rely on having a reliable segmenter. In our experiments, the transients are isolated and the noise level is low, therefore a simple segmenter is all that is needed. Figure 2. shows a segmenter that we implemented in software and which consists of a three layer neural network.

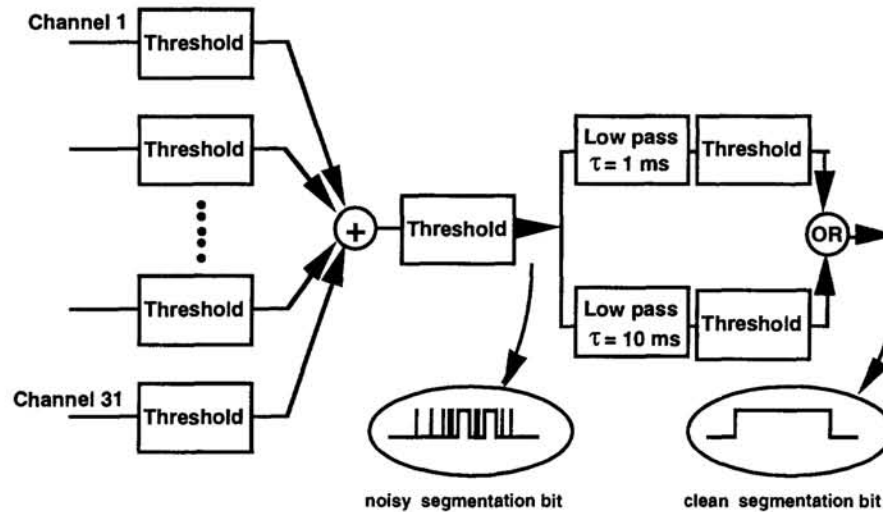

Figure 2: Schematic diagram showing the segmenter network

The input layer receives mean subtracted and rectified signals from the cochlear filters. The first layer simply thresholds these signals. The second layer consists of a single unit that accumulates and rethresholds the thresholded signals. The second layer outputs a noisy segmentation signal that is nonzero if two or more channels in the input layer exceed the input threshold. Finally, the output neuron cleans up the segmentation signal by low-pass filtering it with a time-scale of 10 ms (to fill in drop outs) and by low-pass filtering it with a time-scale of 1 ms (to catch the onset of a transient). The outputs of the two low-pass filters are OR'ed by the output neuron to produce a clean segmentation bit.

The four adjustable thresholds in the network were determined empirically so as to maximize the number of true transients that were properly segmented while minimizing the number of transients that were missed or cut in half.

***C. Smoothing & Normalization:*** The raw output of the filter bank is rectified and smoothed with a single pole filter and subsequently normalized. Smoothing was done with a the

same time-scale (1-ms) in all frequency channels. Let $\mathbf{X}(t)$ be the instantaneous vector of rectified and smoothed channel data, then the instantaneous output of the normalizer is

$\hat{\mathbf{X}}(t) = \dfrac{\mathbf{X}(t)}{\theta + \|\mathbf{X}(t)\|}$. Where $\theta$ is a positive constant whose purpose is to prevent the

normalization stage from amplifying noise in the absence of a transient signal. With this normalization we have $\left\|\hat{\mathbf{X}}(t)\right\|_1 \approx 0$ if $\|\mathbf{X}(t)\|_1 << \theta$, and $\left\|\hat{\mathbf{X}}(t)\right\|_1 \approx 1$ if $\|\mathbf{X}(t)\|_1 >> \theta$. Thus $\theta$ effectively determines a soft input threshold that transients must exceed if they are to be normalized and passed on to higher level processing.

A sequence of normalized vectors over a time-window of length T is used as the feature vector for the correlation and classification stages of the algorithm. Figure 3. shows four normalized feature vectors from one class of transients (concatenated together) .

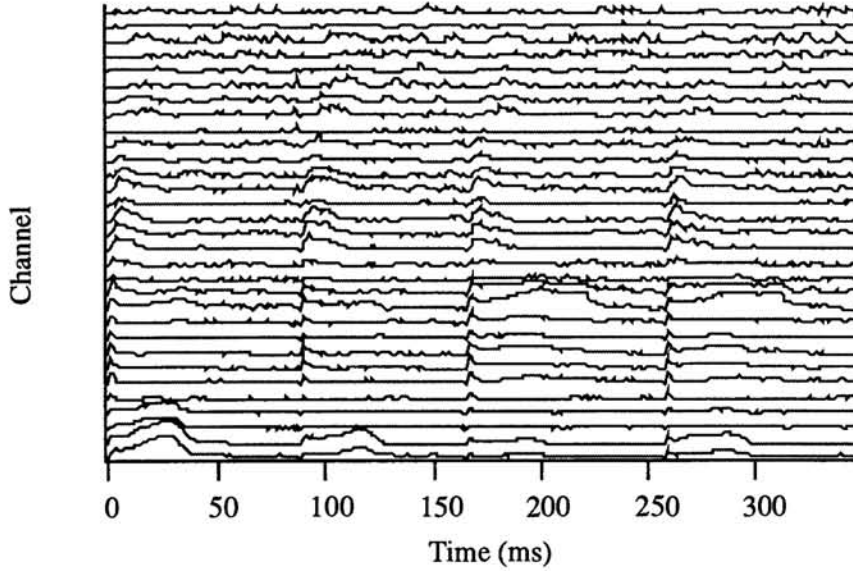

Figure 3.: Normalized representation of the first 4 exemplars from one class of transients.

***D. Correlation :*** The feature-vectors are correlated in the time-frequency domain against a set of $K$ time-frequency templates. The $k-th$ feature-vector-template is precalculated by averaging over a corpus of vectors from the $k-th$ class. Thus, if $C_k$ represents the $k-th$ transient class, and if $\langle \ \rangle_k$ represents an average over the elements in a class, e.g. $\left\langle \hat{\mathbf{X}}(t) \right\rangle_k = E\left\{ \hat{\mathbf{X}}(t) \middle| \hat{\mathbf{X}}(t) \in C_k \right\}$. Then the template is of the form $b_k(t) = \left\langle \hat{\mathbf{X}}(t) \right\rangle_k$. The instantaneous output of the correlation stage is a $K$-dimensional vector $\mathbf{c}(t)$ whose $k-th$ component is $c_k(t) = \sum\limits_{t'=t-T}^{t} \hat{\mathbf{X}}(t) \cdot \mathbf{b}_k(t)$. The time-frequency window over which the correlations are performed is of length $T$ and is advanced by one time-step between correlation calculations.

***E. Classification*** The classification stage is a simple winner-take-all algorithm that assigns a class to the feature vector by picking the component of $c_k(t)$ that has the largest value at the appropriate time, i.e. $class = \arg\max\limits_{k}\{c_k(t_{valid})\}$.

The segmenter is used to determine the time $t_{valid}$ when the output of the winner-take-all is to be used for classification. This corresponds to properly aligning the feature vector and the template. Leave-one-out cross-validation was used to estimate the out-of-sample classification performance of all the algorithms described in this paper. The rate of correct classification for the baseline algorithm was 92.8%. Out of a total of 221 events that were detected and segmented, 16 were misclassified.

## 3  A CORRELATION ALGORITHM FOR ANALOG VLSI

We now address the question of how to perform classification without performing analog-analog multiplication and without having to store analog templates. To provide a better understanding of the algorithm, we present it as a set of incremental modifications to the baseline algorithm. This will serve to make clear the role played by each modification.

Examination of the normalized representation in figure 3 suggests that the information content of any one time-frequency bin cannot be very high. Accordingly, we seek a highly compressed representation that is both easy to form and with which it is easy to compute. As a preliminary step to forming this compressed representation, consider correlating the time-derivative of the feature vector with the time-derivative of the template,

$$c_k(t) = \sum_{t'=t-T}^{t} \dot{\mathbf{X}}(t) \cdot \mathbf{b}_k(t) \quad \text{where} \quad b_k(t) = \left\langle \dot{\mathbf{X}}(t) \right\rangle_k .$$

This modification has no effect on the out-of-sample performance of the winner-take-all classification algorithm. The above representation, by itself, has very few implementation advantages. It can, in principal, mitigate the effect of any systematic offsets that might emerge from the normalization circuit. Unfortunately, the price for this small advantage would be a very complex multiplier. This is evident since the time-derivative of a positive quantity can have either sign, both the feature vector and the template are now bipolar. Accordingly the correlation hardware would now require 4-quadrant analog-analog multipliers. Moreover the storage circuits must handle bipolar quantities as well.

The next step in forming a compressed representation is to replace the time-differentiated *template* with just a sign that indicates whether the template value in a particular channel is increasing or decreasing with time. This template is $b'_k(t) = Sign\left(\left\langle \dot{\mathbf{X}}(t) \right\rangle_k\right)$. We denote this template as the [-1,+1]-representation template. The resulting classification algorithm yields *exactly* the same out-of-sample performance as the baseline algorithm. The 4-quadrant analog-analog multiply of the differentiated representation is reduced to a "4-quadrant analog-binary" multiply. The storage requirements are reduced to a single bit per time-frequency bin. To simplify the hardware yet further, we exploit the fact that the time derivative of a random unit vector $\mathbf{u}(t)$ (with respect to the 1-norm) satisfies

$$E\left\{ \sum_v Sign(\langle \dot{u}_v \rangle) \dot{u}_v \right\} = 2E\left\{ \sum_v \Theta(\langle \dot{u}_v \rangle) \dot{u}_v \right\}$$

where $\Theta$ is a step function. Accordingly, if we use a template whose elements are in [0,1] instead of [-1, +1], i.e. $b''_k(t) = \Theta\left(\left\langle \dot{\mathbf{X}}(t) \right\rangle_k\right)$, we expect

$E\left\{ \sum_v b'_v \dot{X}_v \right\} = 2E\left\{ b''_v \dot{X}_v \right\} = \left\| \dot{\mathbf{X}} \right\|_1$, provided the feature vector $\dot{\mathbf{X}}(t)$ is drawn from the

same class as is used to calculate the template. Furthermore, if the feature vector and the template are statistically independent, then we expect that either representation will produce a zero correlation, $E\left\{\sum_v b'_v \hat{\dot{X}}_v\right\} = E\left\{b''_v \hat{\dot{X}}_v\right\} = 0$. In practice, we find that the difference in correlation values between using the [0,1] and the [-1,+1] representations is simply a scale factor (approximately equal to 2 to several digits of precision). This holds even when the feature vectors and the templates do not correspond to the same class. Thus the difference between the two representations is quantitatively minor and qualitatively nonexistent, as evidenced by our classification experiments, which show that the out-of-sample performance of the [0,1] representation is *identical* to that of the [-1,+1] representation. Furthermore, changing to the [0,1] representation has no impact on the storage requirements since both representations require the storage of single bit per time-frequency bin. On the other hand, consider that by using the [0,1] representation we now have a "2-quadrant analog-binary" multiply instead of a "4-quadrant analog-binary" multiply. Finally, we observe that differentiation and correlation are commuting operations, thus rather than differentiating $\hat{X}(t)$ before correlation, we can differentiate after the correlation without changing the result. This reduces the complexity of the correlation operation still further, since the fact that both $\hat{X}(t)$ and $b''_k(t)$ are positive means that we need only implement a correlator with 1-quadrant analog-binary multiplies.

The result of the above evolution is a correlation algorithm that empirically performs as well as a baseline correlation algorithm, but only requires binary-multiplexing to perform the correlation. We find that with only 16 frequency channels and 64 time bins (1024-bits/templates) , we are able to achieve the desired level of performance. We have undertaken the design and fabrication of a prototype chip. This chip has been fabricated and we will report on it's performance in the near future. Figure 4 illustrates the key architectural features of the correlator/memory implementation. The rectified and

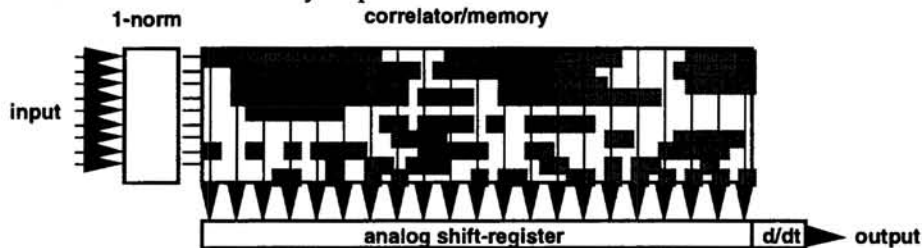

Figure 4: Schematic architecture of the k-th correlator-memory.

smoothed frequency-analyzed signals are input from the left as currents. The currents are normalized before being fed into the correlator. A binary time-frequency template is stored as a bit pattern in the correlator/memory. A single bit is stored at each time and frequency bin. If this bit is set, current is mirrored from the horizontal (frequency) lines onto vertical (aggregation) lines. Current from the aggregation lines is integrated and shifted in a bucket-brigade analog shift register. The last two stages of the shift register are differenced to estimate a time-derivative.

# 4 DISCUSSION AND CONCLUSIONS

The correlation algorithm described in the previous section is related to the zero-crossing

representation analyzed by Yang, Wang. and Shamma (1992). This is because bit flips in the templates correspond to the zero crossings of the expected time-derivative of the normalized "energy-envelope." Note that we do not encode the incoming acoustic signal with a zero-crossing representation. Interestingly enough, if both the analog signal *and* the template are reduced to a binary representation, then the classification performance drops dramatically. It appears that maintaining some analog information in the processing path is significant.

The frequency-domain normalization approach presented above throws away absolute intensity information. Thus, low intensity resonances that remain excited after the initial burst of acoustic energy are as important in the feature vector as the initial burst of energy. These resonances can contain significant information about the nature of the transient but would have less weight in an algorithm with a different normalization scheme. Another consequence of the normalization is that even a transient whose spectrum is highly concentrated in just a few frequency channels will spread its information over the entire spectrum through the normalization denominator. The use of a normalized representation thus distributes the correlation calculation over very many frequency channels and serves to mitigate the effect of device mismatch.

We consider the proposed correlator/memory as a potential component in more sophisticated acoustic processing systems. For example, the continuously generated output of the correlators , $c(t)$, is itself a feature vector that could be used in more sophisticated segmentation and/or classification algorithms such as the time-delayed neural network approach of Unnikrishnan, Hopfield and Tank (1991).

The work reported in this report was supported by a Whiting School of Engineering/Applied Physics Laboratory Collaborative Grant. Preliminary work was supported by an APL Internal Research & Development Budget.

# REFERENCES

Furth , P.M. and Kumar, N.G., Andreou, A.G. and Goldstein, M.H. , "Experiments with the Hopkins Electronic EAR", 14th Speech Research Symposium, Baltimore, MD pp.183-189, (1994).

Pineda, F.J., Ryals, K., Steigerwald, D. and Furth, P., (1995). "Acoustic Transient Processing using the Hopkins Electronic Ear", World Conference on Neural Networks 1995, Washington DC.

Yang, X., Wang K. and Shamma, S.A. (1992). "Auditory Representations of Acoustic Signals", IEEE Trans. on Information Processing, 38, pp. 824-839.

Morgan, N. , Bourlard, H., Greenberg, S., Hermansky, H. and Wu, S. L., (1996). "Stochastic Perceptual Models of Speech", IEEE Proc. Intl. Conference on Acoustics, Speech and Signal Processing, Detroit, MI, pp. 397-400.

Unnikrishnan, K.P., Hopfield J.J., and Tank, D.W. (1991). "Connected-Digit Speaker-Dependent Speech Recognition Using a Neural Network with Time-Delayed Connections", IEEE Transactions on Signal Processing, 39, pp. 698-713